# Audio-Vision:
# Using Audio-Visual Synchrony to Locate Sounds

**John Hershey** *
jhershey@cogsci.ucsd.edu
Department of Cognitive Science
University of California, San Diego
La Jolla, CA 92093-0515

**Javier Movellan**
movellan@cogsci.ucsd.edu
Department of Cognitive Science
University of California, San Diego
La Jolla, CA 92093-0515

## Abstract

Psychophysical and physiological evidence shows that sound localization of acoustic signals is strongly influenced by their synchrony with visual signals. This effect, known as ventriloquism, is at work when sound coming from the side of a TV set feels as if it were coming from the mouth of the actors. The ventriloquism effect suggests that there is important information about sound location encoded in the synchrony between the audio and video signals. In spite of this evidence, audiovisual synchrony is rarely used as a source of information in computer vision tasks. In this paper we explore the use of audio visual synchrony to locate sound sources. We developed a system that searches for regions of the visual landscape that correlate highly with the acoustic signals and tags them as likely to contain an acoustic source. We discuss our experience implementing the system, present results on a speaker localization task and discuss potential applications of the approach.

## Introduction

We present a method for locating sound sources by sampling regions of an image that correlate in time with the auditory signal. Our approach is inspired by psychophysical and physiological evidence suggesting that audio-visual contingencies play an important role in the localization of sound sources: sounds seem to emanate from visual stimuli that are synchronized with the sound. This effect becomes particularly noticeable when the perceived source of the sound is known to be false, as in the case of a ventriloquist's dummy, or a television screen. This phenomenon is known in the psychophysical community as the *ventriloquism effect*, defined as a mislocation of sounds toward their apparent visual source. The effect is robust in a wide variety of conditions, and has been found to be strongly dependent on the degree of "synchrony" between the auditory and visual signals (Driver, 1996; Bertelson, Vroomen, Wiegeraad & de Gelder, 1994).

---

To whom correspondence should be addressed.

The ventriloquism effect is in fact less speech-specific than first thought. For example the effect is not disrupted by an upside-down lip signal (Bertelson, Vroomen, Wiegeraad & de Gelder, 1994) and is just as strong when the lip signals are replaced by light flashes that are synchronized with amplitude peaks in the audio signal (Radeau & Bertelson, 1977). The crucial aspect here is correlation between visual and auditory intensity over time. When the light flashes are not synchronized the effect disappears.

The ventriloquism effect is strong enough to produce an enduring localization bias, known as the *ventriloquism aftereffect*. Over time, experience with spatially offset auditory-visual stimuli causes a persistent shift in subsequent auditory localization. Exposure to audio-visual stimuli offset from each other by only 8 degrees of azimuth for 20-30 minutes is sufficient to shift auditory localization by the same amount. A corresponding shift in neural processing has been detected in macaque monkeys as early as primary auditory cortex(Recanzone, 1998). In barn owls a misalignment of visual and auditory stimuli during development causes the realignment of the auditory and visual maps in the optic tectum(Zheng & Knudsen, 1999; Stryker, 1999; Feldman & Knudsen, 1997).

The strength of the psychophysical and physiological evidence suggests that audio-visual contingency may be used as an important source of information that is currently underutilized in computer vision tasks. Visual and auditory sensor systems carry information about the same events in the world, and this information must be combined correctly in order for a useful interaction of the two modalities. Audiovisual contingency can be exploited to help determine which signals in different modalities share a common origin. The benefits are two-fold: the two signals can help localize each other, and once paired can help interpret each other. To this effect we developed a system to localize speakers using input from a camera and a single microphone. The approach is based on searching for regions of the image which are "synchronized" with the acoustic signal.

## Measuring Synchrony

The concept of audio-visual *synchrony* is not well formalized in the psychophysical literature, so for a working definition we interpret synchrony as the degree of mutual information between audio and spatially localized video signals. Ultimately it is a *causal* relationship that we are often interested in, but causes can only be inferred from effects such as synchrony. Let $a(t) \in \mathbb{R}^n$ be a vector describing the acoustic signal at time $t$. The components of $a(t)$ could be cepstral coefficients, pitch measurements, or the outputs of a filter bank. Let $v(x, y, t) \in \mathbb{R}^m$ be a vector describing the visual signal at time $t$, pixel $(x, y)$. The components of $v(x, y, t)$ could represent Gabor energy coefficients, RGB color values, etc.

Consider now a set of $s$ audio and visual vectors $\mathcal{S} = (a(t_l), v(x, y, t_l))_{l=k-s-1,\cdots,k}$ sampled at times $t_{k-s-1}, \cdots, t_k$ and at spatial coordinates $(x, y)$. Given this set of vectors our goal is to provide a number that describes the temporal contingency between audio and video at time $t_k$. The approach we take is to consider each vector in $\mathcal{S}$ as an independent sample from a joint multivariate Gaussian process $(A(t_k), V(x, y, t_k))$ and define audio-visual synchrony at time $t_k$ as the estimate of the mutual information between the audio and visual components of the process.

Let $A(t_k) \sim \mathcal{N}_n(\mu_A(t_k), \Sigma_A(t_k))$, and $V(x, y, t_k) \sim \mathcal{N}_m(\mu_V(x, y, t), \Sigma_V(x, y, t_k))$, where $\mu$ represents means and $\Sigma$ covariance matrices. Let $A(t_k)$ and $V(x, y, t_k)$ be jointly Gaussian, i.e., $(A(t_k), V(x, y, t_k)) \sim \mathcal{N}_{n+m}(\mu_{A,V}(x, y, t_k), \Sigma_{A,V}(x, y, t_k))$.

The mutual information between $A(x, y, t_k)$ and $V(t_k)$ can be shown to be as follows

$$
\begin{aligned}
I(A(t_k); V(x, y, t_k)) &= H(A(t_k)) + H(V(x, y, t_k)) - H(A(t_k), V(x, y, t_k)) \\
&= \frac{1}{2} \log(2\pi e)^n |\Sigma_A(t_k)| + \frac{1}{2} \log(2\pi e)^m |\Sigma_V(x, y, t_k)| \quad (1) \\
&\quad - \frac{1}{2} \log(2\pi e)^{n+m} |\Sigma_{A,V}(x, y, t_k)|
\end{aligned}
$$

$$(2)$$

$$
= \frac{1}{2} \log \frac{|\Sigma_A(t_k)||\Sigma_V(x, y, t_k)|}{|\Sigma_{A,V}(x, y, t_k)|}. \quad (3)
$$

In the special case that $n = m = 1$, then

$$
I(A(t_k); V(x, y, t_k)) = -\frac{1}{2} \log(1 - \rho^2(x, y, t_k)), \quad (4)
$$

where $\rho(x, y, t_k)$ is the Pearson correlation coefficient between $A(t_k)$ and $V(x, y, t_k)$.

For each triple $(x, y, t_k)$ we estimate the mutual information between $A(t_k)$ and $V(x, y, t_k)$ by considering each element of $S$ as an independent sample from the random vector $(A(t_k), V(x, y, t_k))$. This amounts to computing estimates of the joint covariance matrix $\Sigma_{A,V}(x, y, t_k)$. For example the estimate of the covariance between the $i^{th}$ audio component and the $j^{th}$ video component would be as follows

$$
S_{A_i, V_j}(x, y, t_k) = \frac{1}{s-1} \sum_{l=0}^{s-1} (a_i(t_{k-l}) - \bar{a}_i(t_k))(v_j(x, y, t_{k-l}) - \bar{v}_j(x, y, t_k)), \quad (5)
$$

where

$$
\bar{a}_i(t_k) = \frac{1}{s} \sum_{l=0}^{s-1} a_i(t_{k-l}), \quad (6)
$$

$$
\bar{v}_j(t_k) = \frac{1}{s} \sum_{l=0}^{s-1} v_j(x, y, t_{k-l}). \quad (7)
$$

$$(8)$$

These simple covariance estimates can be computed recursively in constant time with respect to the number of timepoints. The independent treatment of pixels would lend well to a parallel implementation.

To measure performance, a secondary system produces a single estimate of the auditory location, for use with a database of labeled solitary audiovisual sources. Unfortunately there are many ways of producing such estimates so it becomes difficult to separate performance of the measure from the underlying system. The model used here is a centroid computation on the mutual information estimates, with some enhancements to aid tracking and reduce background noise.

## Implementation Issues

A real time system was prototyped using a QuickCam on the Linux operating system and then ported to NT as a DirectShow filter. This platform provides input from real-time audio and video capture hardware as well as from static movie files. The video output could also be rendered live or compressed and saved in a movie file. The implementation was challenging in that it turns out to be rather difficult

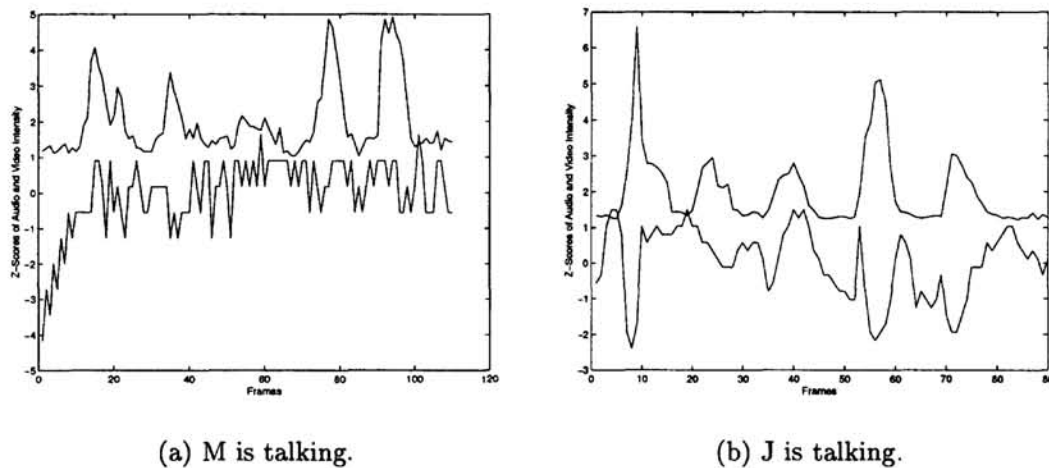

(a) M is talking.                                                    (b) J is talking.

Figure 1: Normalized audio and visual intensity across sequences of frames in which a sequence of four numbers is spoken. The top trace is the contour of the acoustic energy from one of two speakers, M or J, and the bottom trace is the contour of intensity values for a single pixel, (147,100), near the mouth of J.

to process precisely time-synchronized audio and video on a serial machine in real time. Multiple threads are required to read from the peripheral audio and visual devices. By the time the audio and visual streams reach the AV filter module, they are quite separate and asynchronous. The separately threaded auditory and visual packet streams must be synchronized, buffered, and finally matched and aligned by time-stamps before they can finally be processed. It is interesting that successful biologial audiovisual systems employ a parallel architecture and thus avoid this problem.

## Results

To obtain a performance baseline we first tried the simplest possible approach: A single audio and visual feature per location: $n = m = 1$, $v(x, y, t) \in \mathbb{R}$ is the intensity of pixel $(x, y)$ at time $t$, and $a(t) \in \mathbb{R}$ is the average acoustic energy over the interval $[t - \Delta t, t]$, where $\Delta t = 1/30 \, msec$, the sampling period for the NTSC video signal. Figure 1 illustrates the time course of these signals for a non-synchronous and a synchronous pair of acoustic energy and pixel intensity. Notice in particular that in the synchonous pair, 1(b), where the sound and pixel values come from the same speaker, the relationship between the signals changes over time. There are regions of positive and negative covariance strung together in succession. Clearly the relationship over the entire sequence is far from linear. However over shorter time periods a linear relationship looks like a better approximation. Our window size of 16 samples (i.e., $s = 16$ in 5 coincides approximately with this time-scale. Perhaps by averaging over many small windows we can capture on a larger scale what would be lost to the same method applied with a larger window. Of course there is a trade-off in the time-scale between sensitivity to spurious transients, and the response time of the system.

We applied this mutual information measure to all the pixels in a movie, in the spirit of the perceptual maps of the brain. The result is a changing topographic map of audiovisual mutual information. Figure 2 illustrates two snapshots in which

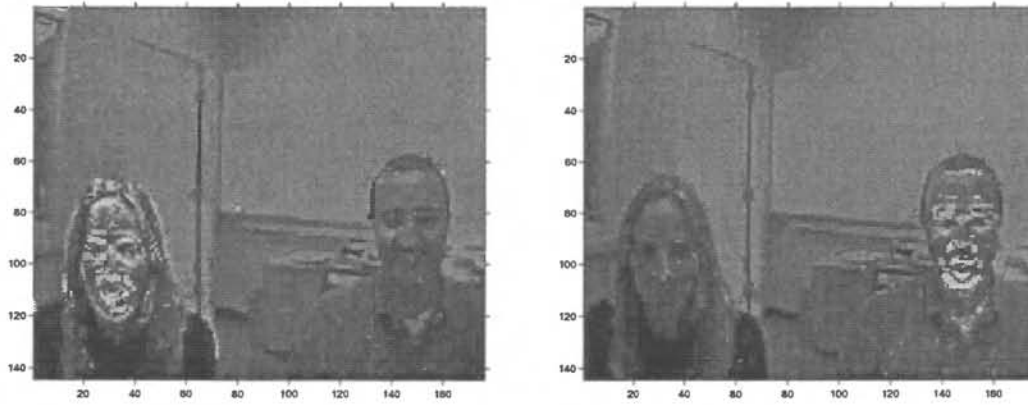

(a) Frame 206: M (at left) is talking.   (b) Frame 104: J (at right) is talking.

Figure 2: Estimated mutual information between pixel intensity and audio intensity (bright areas indicate greater mutual information) overlaid on stills from the video where one person is in mid-utterance.

different parts of the face are synchronous (possibly with different sign) with the sound they take part in producing. It is interesting that the synchrony is shared by some parts, such as the eyes, that do not directly contribute to the sound, but contribute to the communication nonetheless.

To estimate the position of the speaker we computed a centroid were each point was weighted by the estimated mutual information between the corrsponding pixel and the audio signal. At each time step the mutual information was estimated using 16 past frames (i.e., $s = 16$) In order to reduce the intrusion of spurious correlations from competing targets, once a target has been found, we employ a Gaussian *influence function*. (Goodall, 1983) The influence function reduces the weight given to mutual information from locations far from the current centroid when computing the next centroid. To allow for the speedy disengagement from a dwindling source of mutual information we set a threshold on the mutual information. Measurements under the threshold are treated as zero. This threshold also reduces the effects of unwanted background noise, such as camera and microphone jitter.

$$\hat{S}_x(t) = \frac{\sum_x \sum_y x\, \theta(\log(1 - \hat{\rho}^2(x,y,t)))\psi(x,\hat{S}_x(t-1))}{\sum_x \sum_y \theta(\log(1 - \hat{\rho}^2(x,y,t)))\psi(x,\hat{S}_x(t-1))} \qquad (9)$$

where $\hat{S}_x(t)$ represents the estimate of the $x$ coordinate for the position of the speaker at time $t$. $\theta(.)$ is the thresholding function, and $\psi(x,\hat{S}_x(t-1))$ is the influence function, which depends upon the position $x$ of the pixel being sampled and the prior estimate $\hat{S}_x(t-1)$. $\hat{\rho}^2(x,y,t)$ is the estimate of the correlation between the intensity in pixel $(x,y)$ and the acoustic enery, when using the 16 past video frames. $-\frac{1}{2}\log(1 - \hat{\rho}^2(x,y,t))$ is the corresponding estimate of mutual information (the factor, $-\frac{1}{2}$ cancels out in the quotient after adjusting the threshold function accordingly.)

We tried the approach on a movie of two people (M and J) taking turns while saying random digits. Figure 3 shows the estimates of the actual positions of the speaker

as a function of time. The estimates clearly provide information that could be used to localize the speaker, especially in combination with other approaches (e.g., flesh detection).

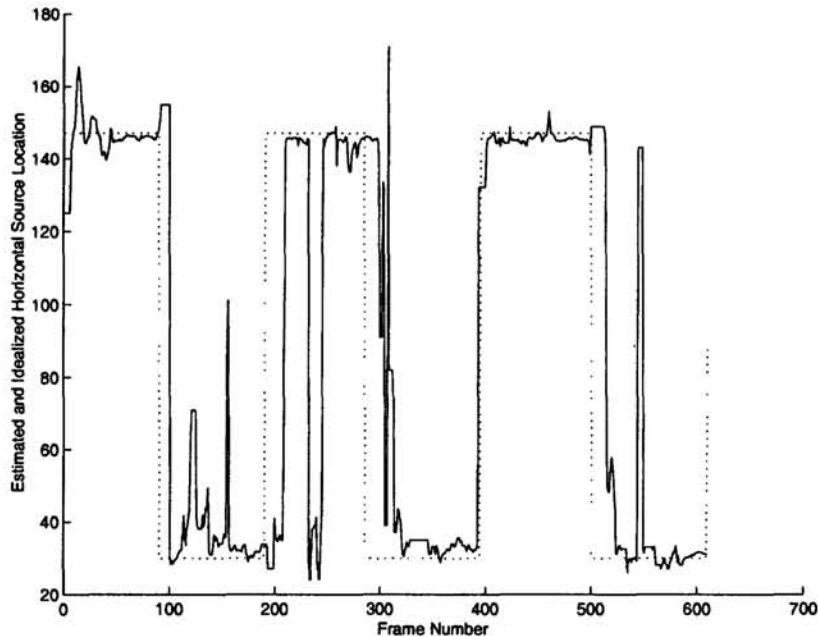

Figure 3: Estimated and actual position of speaker at each frame for six hundred frames. The sources, M and J, took turns uttering a series of four digits, for three turns each. The actual positions and alternation times were measured by hand from the video recording

## Conclusions

We have presented exploratory work on a system for localizing sound sources on a video signal by tagging regions of the image that are correlated in time with the auditory signal. The approach was motivated by the wealth of evidence in the psychophysical and physiological literature showing that sound localization is strongly influenced by synchrony with the visual signal. We presented a measure of local synchrony based on modeling the audio-visual signal as a non-stationary Gaussian process. We developed a general software tool that accepts as inputs all major video and audio file formats as well as direct input from a video camera. We tested the tool on a speaker localization task with very encouraging results. The approach could have practical applications for localizing sound sources in situations where where acoustic stereo cues are inexistent or unreliable. For example the approach could be used to help localize the actor talking in a video scene and put closed-captioned text near the audio source. The approach could also be used to guide a camera in teleconferencing applications.

While the results reported here are very encouraging, more work needs to be done before practical applications are developed. For example we need to investigate more sophisticated methods for processing the audio and video signals. At this point we use average energy to represent the video and thus changes in the fundamental frequency that do not affect the average energy would not be captured by our model. Similarly local video decompositions, like spatio-temporal Gabor filtering, or approaches designed to enhance the lip regions may be helpful. The

changing symmetry observed between audio and video signals might be addressed rectifying or squaring the normalized signals and derivatives. Finally, relaxing the Gaussian constraints in our measure of audio-visual contingency may help improve performance. While the work shown here is exploratory at this point, the approach is very promising: It emphasizes the idea of machine perception as a multimodal process it is backed by psychophysical evidence, and when combined with other approaches it may help improve robustness in tasks such as localization and separation of sound sources.

## References

Bertelson, P., Vroomen, J., Wiegeraad, G., & de Gelder, B. (1994). Exploring the relation between McGurk interference and ventriloquism. In *Proceedings of the 1994 International Conference on Spoken Language Processing*, volume 2, pages 559–562.

Driver, J. (1996). Enhancement of selective listening by illusory mislocation of speech sounds due to lip-reading. *Nature, 381*, 66–68.

Feldman, D. E. & Knudsen, E. I. (1997). An anatomical basis for visual calibration of the auditiory space map in the barn owl's midbrain. *The Journal of Neuroscience, 17*(17), 6820–6837.

Goodall, C. (1983). M-Estimators of Location: an outline of the theory. Wiley series in probability and mathematical statistics. Applied probability and statistics.

Radeau, M. & Bertelson, P. (1977). Adaptation to auditory-visual discordance and ventriloquism in semi-realistic situations. *Perception and Psychophysics, 22*, 137–146.

Recanzone, G. H. (1998). Rapidly induced auditory plasticity: The ventriloquism aftereffect. *Proceedings of the National Academy of Sciences, USA, 95*, 869–875.

Stryker, M. P. (1999). Sensory Maps on the Move. *Science*, 925–926.

Zheng, W. & Knudsen, E. I. (1999). Functional Selection of Adaptive Auditory Space Map by GABAA-Mediated Inhibition, 962–965.
